# Convergence Rates of Algorithms for Visual Search: Detecting Visual Contours

**A.L. Yuille**
Smith-Kettlewell Inst.
San Francisco, CA 94115

**James M. Coughlan**
Smith-Kettlewell Inst.
San Francisco, CA 94115

## Abstract

This paper formulates the problem of visual search as Bayesian inference and defines a Bayesian ensemble of problem instances. In particular, we address the problem of the detection of visual contours in noise/clutter by optimizing a global criterion which combines local intensity and geometry information. We analyze the convergence rates of A* search algorithms using results from information theory to bound the probability of rare events within the Bayesian ensemble. This analysis determines characteristics of the domain, which we call order parameters, that determine the convergence rates. In particular, we present a specific admissible A* algorithm with pruning which converges, with high probability, with expected time $O(N)$ in the size of the problem. In addition, we briefly summarize extensions of this work which address fundamental limits of target contour detectability (i.e. algorithm independent results) and the use of non-admissible heuristics.

## 1   Introduction

Many problems in vision, such as the detection of edges and object boundaries in noise/clutter, see figure (1), require the use of search algorithms. Though many algorithms have been proposed , see Yuille and Coughlan (1997) for a review, none of them are clearly optimal and it is difficult to judge their relative effectiveness. One approach has been to compare the results of algorithms on a representative dataset of images. This is clearly highly desirable though determining a representative dataset is often rather subjective.

In this paper we are specifically interested in the convergence rates of A* algorithms (Pearl 1984). It can be shown (Yuille and Coughlan 1997) that many algorithms proposed to detect visual contours are special cases of A* . We would like to understand what characteristics of the problem domain determine the convergence

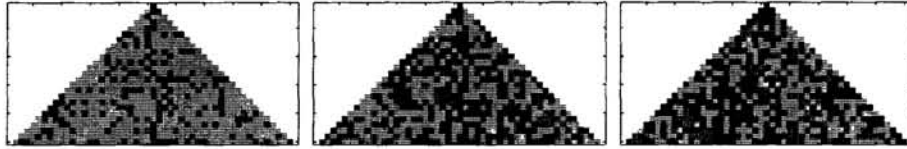

Figure 1: The difficulty of detecting the target path in clutter depends, by our theory (Yuille and Coughlan 1998), on the order parameter $K$. The larger $K$ the less computation required. Left, an easy detection task with $K = 3.1$. Middle, a hard detection task $K = 1.6$. Right, an impossible task with $K = -0.7$.

rates.

We formulate the problem of detecting object curves in images to be one of statistical estimation. This assumes statistical knowledge of the images and the curves, see section (2). Such statistical knowledge has often been used in computer vision for determining optimization criteria to be minimized. We want to go one step further and use this statistical knowledge to determine good search strategies by defining a *Bayesian ensemble* of problem instances. For this ensemble, we can prove certain curve and boundary detection algorithms, with high probability, achieve expected time convergence in time linear with the size of the problem. Our analysis helps determine important characteristics of the problem, which we call *order parameters*, which quantify the difficulty of the problem.

The next section (2) of this paper describes the basic statistical assumptions we make about the domain and describes the mathematical tools used in the remaining sections. In section (3) we specify our search algorithm and establish convergence rates. We conclude by placing this work in a larger context and summarizing recent extensions.

## 2   Statistical Background

Our approach assumes that both the intensity properties and the geometrical shapes of the target path (i.e. the edge contour) can be determined statistically. This path can be considered to be a set of elementary path segments joined together. We first consider the intensity properties along the edge and then the geometric properties. The set of all possible paths can be represented by a tree structure, see figure (2).

The image properties at segments lying on the path are assumed to differ, in a statistical sense, from those off the path. More precisely, we can design a filter $\phi(.)$ with output $\{y_x = \phi(I(x))\}$ for a segment at point $x$ so that:

$$P(y_x) = P_{on}(y_x), \quad if \ ``x" \ lies \ on \ the \ true \ path$$
$$P(y_x) = P_{off}(y_x), \quad if \ ``x" \ lies \ off \ the \ true \ path. \qquad (1)$$

For example, we can think of the $\{y_x\}$ as being values of the edge strength at point $x$ and $P_{on}, P_{off}$ being the probability distributions of the response of $\phi(.)$ on and off an edge. The set of possible values of the random variable $y_x$ is the *alphabet* with *alphabet size* $M$ (i.e. $y_x$ can take any of $M$ possible values). See (Geman and Jedynak 1996) for examples of distributions for $P_{on}, P_{off}$ used in computer vision applications.

We now consider the geometry of the target contour. We require the path to be made up of connected segments $x_1, x_2, \ldots, x_N$. There will be a Markov probability distribution $P_g(x_{i+1}|x_i)$ which specifies prior probabilistic knowledge of the target.

It is convenient, in terms of the graph search algorithms we will use, to consider that each point $x$ has a set of $Q$ neighbours. Following terminology from graph theory, we refer to $Q$ as the *branching factor*. We will assume that the distribution $P_g$ depends only on the relative positions of $x_{i+1}$ and $x_i$. In other words, $P_g(x_{i+1}|x_i) = P_{\Delta g}(x_{i+1} - x_i)$. An important special case is when the probability distribution is uniform for all branches (i.e. $P_{\Delta g}(\Delta x) = U(\Delta x) = 1/Q, \forall \Delta x$). The joint distribution $P(X, Y)$ of the road geometry $X$ and filter responses $Y$ determines the *Bayesian Ensemble*.

By standard Bayesian analysis, the optimal path $X^* = \{x_1^*, \ldots, x_N^*\}$ maximizes the sum of the log posterior:

$$E(X) = \sum_i \log \frac{P_{on}(y_{(x_i)})}{P_{off}(y_{(x_i)})} + \sum_i \log \frac{P_{\Delta g}(x_{i+1} - x_i)}{U(x_{i+1} - x_i)}, \qquad (2)$$

where the sum $i$ is taken over all points on the target. $U(x_{i+1} - x_i)$ is the uniform distribution and its presence merely changes the log posterior $E(X)$ by a constant value. It is included to make the form of the intensity and geometric terms similar, which simplifies our later analysis.

We will refer to $E(X)$ as the *reward* of the path $X$ which is the sum of the *intensity rewards* $\log \frac{P_{on}(y_{(x_i)})}{P_{off}(y_{(x_i)})}$ and the *geometric rewards* $\log \frac{P_{\Delta g}(x_{i+1}-x_i)}{U(x_{i+1}-x_i)}$.

It is important to emphasize that our results can be extended to higher-order Markov chain models (provided they are shift-invariant). We can, for example, define the $x$ variable to represent spatial orientation *and* position of a small edge segment. This will allow our theory to apply to models, such as snakes, used in recent successful vision applications (Geman and Jedynak 1996). (It is straightforward to transform the standard energy function formulation of snakes into a Markov chain by discretizing and replacing the derivatives by differences. The smoothness constraints, such as membranes and thin plate terms, will transform into first and second order Markov chain connections respectively). Recent work by Zhu (1998) shows that Markov chain models of this type can be learnt using Minimax Entropy Learning theory from a representative set of examples. Indeed Zhu goes further by demonstrating that other Gestalt grouping laws can be expressed in this framework and learnt from representative data.

Most Bayesian vision theories have stopped at this point. The statistics of the problem domain are used only to determine the optimization criterion to be minimized and are not exploited to analyze the complexity of algorithms for performing the optimization. In this paper, we go a stage further. We use the statistics of the problem domain to define a Bayesian ensemble and hence to *determine the effectiveness of algorithms for optimizing* criteria such as (2). To do this requires the use of Sanov's theorem for calculating the probability of rare events (Cover and Thomas 1991). For the road tracking problem this can be re-expressed as the following theorem, derived in (Yuille and Coughlan 1998):

**Theorem 1.** *The probabilities that the spatially averaged log-likelihoods on, and off, the true curve are above, or below, threshold $T$ are bounded above as follows:*

$$Pr\{\frac{1}{n} \sum_{i=1}^n \{\log \frac{P_{on}(y_{(x_i)})}{P_{off}(y_{(x_i)})}\}_{on} < T\} \le (n+1)^M 2^{-nD(P_T \| P_{on})} \qquad (3)$$

$$Pr\{\frac{1}{n} \sum_{i=1}^n \{\log \frac{P_{on}(y_{(x_i)})}{P_{off}(y_{(x_i)})}\}_{off} > T\} \le (n+1)^M 2^{-nD(P_T \| P_{off})}, \qquad (4)$$

where the subscripts $_{on}$ and $_{off}$ mean that the data is generated by $P_{on}, P_{off}$, $P_T(y) = P_{on}^{1-\lambda(T)}(y)P_{off}^{\lambda(T)}/Z(T)$ where $0 \le \lambda(T) \le 1$ is a scalar which depends on the threshold $T$ and $Z(T)$ is a normalization factor. The value of $\lambda(T)$ is determined by the constraint $\sum_y P_T(y) \log \frac{P_{on}(y)}{P_{off}(y)} = T$.

In the next section, we will use Theorem 1 to determine a criterion for pruning the search based on comparing the intensity reward to a threshold $T$ (pruning will also be done using the geometric reward). The choice of $T$ involves a trade-off. If $T$ is large (i.e. close to $D(P_{on}||P_{off})$) then we will rapidly reject false paths but we might also prune out the target (true) path. Conversely, if $T$ is small (close to $-D(P_{off}||P_{on})$) then it is unlikely we will prune out the target path but we may waste a lot of time exploring false paths. In this paper we choose $T$ large and write the fall-off factors (i.e. the exponents in the bounds of equations (3,4)) as $D(P_T||P_{on}) = \epsilon_1(T)$, $D(P_T||P_{off}) = D(P_{on}||P_{off}) - \epsilon_2(T)$ where $\epsilon_1(T), \epsilon_2(T)$ are positive and $(\epsilon_1(T), \epsilon_2(T)) \mapsto (0,0)$ as $T \mapsto D(P_{on}||P_{off})$. We perform a similar analysis for the geometric rewards by substituting $P_{\Delta g}, U$ for $P_{on}, P_{off}$. We choose a threshold $\hat{T}$ satisfying $-D(U||P_{\Delta g}) < \hat{T} < D(P_{\Delta g}||U)$. The results of Theorem 1 apply with the obvious substitutions. In particular, the alphabet factor becomes $Q$ (the branching factor). Once again, in this paper, we choose $\hat{T}$ to be large and obtain fall-off factors $D(P_{\hat{T}}||P_{\Delta g}) = \hat{\epsilon}_1(\hat{T})$, $D(P_{\hat{T}}||U) = D(P_{\Delta g}||U) - \hat{\epsilon}_2(\hat{T})$.

## 3   Tree Search: A*, heuristics, and block pruning

We now consider a specific example, motivated by Geman and Jedynak (1996), of searching for a path through a search tree. In Geman and Jedynak the path corresponds to a road in an aerial image and they assume that they are given an initial point and direction on the target path. They have a branching factor $Q = 3$ and, in their first version, the prior probability of branching is considered to be the uniform distribution (later they consider more sophisticated priors). They assume that no path segments overlap which means that the search space is a tree of size $Q^N$ where $N$ is the size of the problem (i.e. the longest length). The size of the problem requires an algorithm that converges in $O(N)$ time and they demonstrate an algorithm which empirically performs at this speed. But no proof of convergence rates are given in their paper. It can be shown, see (Yuille and Coughlan 1997), that the Geman and Jedynak algorithm is a close approximation to A* which uses pruning. (Observe that Geman and Jedynak's tree representation is a simplifying assumption of the Bayesian model which assumes that once a path diverges from the true path it can never recover, although we stress that the *algorithm* is able to recover from false starts – for more details see Coughlan and Yuille 1998).

We consider an algorithm which uses an admissible A* heuristic and a pruning mechanism. The idea is to examine the paths chosen by the A* heuristic. As the length of the candidate path reaches an integer multiple of $N_0$ we prune it based on its intensity reward and its geometric reward evaluated on the previous $N_0$ segments, which we call a *segment block*. The reasoning is that few false paths will survive this pruning for long but the target path will survive with high probability.

We prune on the intensity by eliminating all paths whose intensity reward, averaged over the last $N_0$ segments, is below a threshold $T$ (recall that $-D(P_{off}||P_{on}) < T < D(P_{on}||P_{off})$ and we will usually select $T$ to take values close to $D(P_{on}||P_{off})$). In addition, we prune on the geometry by eliminating all paths whose geometric rewards, averaged over the last $N_0$ segments, are below $\hat{T}$ (where $-D(U||P_{\Delta g}) < \hat{T} < D(P_{\Delta g}||U)$ with $\hat{T}$ typically being close to $D(P_{\Delta g}||U)$). More precisely, we

discard a path provided (for any integer $z \geq 0$):

$$\frac{1}{N_0} \sum_{i=zN_0+1}^{(z+1)N_0} \log \frac{P_{on}(y_i)}{P_{off}(y_i)} < T, \ or \ \frac{1}{N_0} \sum_{i=zN_0+1}^{(z+1)N_0} \log \frac{P_{\Delta g}(\Delta x_i)}{U(\Delta x_i)} < \hat{T}. \tag{5}$$

There are two important issues to address: (i) With what probability will the algorithm converge?, (ii) How long will we expect it take to converge? The next two subsections put bounds on these issues.

## 3.1 Probability of Convergence

Because of the pruning, there is a chance that there will be no paths which survive pruning. To put a bound on this we calculate the probability that the target (true) path survives the pruning. This gives a lower bound on the probability of convergence (because there could be false paths which survive even if the target path is mistakenly pruned out).

The pruning rules removes path segments for which the intensity reward $r_I$ or the geometric reward $r_g$ fails the pruning test. The probability of failure by removing a block segment of the true path, with rewards $r_I^t, r_g^t$, is $Pr(r_I^t < T \ or \ r_g^t < \hat{T}) \leq Pr(r_I^t < T) + Pr(r_g^t < \hat{T}) \leq (N_0 + 1)^M 2^{-N_0 \epsilon_1(T)} + (N_0 + 1)^Q 2^{-N_0 \hat{\epsilon}_1(\hat{T})}$, where we have used Theorem 1 to put bounds on the probabilities. The probability of pruning out any $N_0$ segments of the true path can therefore be made arbitrarily small by choosing $N_0, T, \hat{T}$ so as to make $N_0 \epsilon_1$ and $N_0 \hat{\epsilon}_1$ large.

It should be emphasized that the algorithm will not necessarily converge to the exact target path. The admissible nature of the heuristic means that the algorithm will converge to the path with highest reward which has survived the pruning. It is highly probable that this path is close to the target path. Our recent results (Coughlan and Yuille 1998, Yuille and Coughlan 1998) enable us to quantify this claim.

## 3.2 Bounding the Number of False Paths

Suppose we face a Q-nary tree. We can order the false paths by the stage at which they diverge from the target (true) path, see figure (2). For example, at the first branch point the target path lies on only one of the $Q$ branches and there are $Q - 1$ false branches which generate the first set of false paths $F_1$. Now consider all the $Q - 1$ false branches at the second target branch, these generate set $F_2$. As we follow along the true path we keep generating these false sets $F_i$. The set of all paths is therefore the target path plus the union of the $F_i$ ($i = 1, \ldots, N$). To determine convergence rates we must bound the amount of time we spend searching the $F_i$. If the expected time to search each $F_i$ is constant then searching for the target path will at most take *constant* $\cdot N$ steps.

Consider the set $F_i$ of false paths which leave the true path at stage $i$. We will apply our analysis to block segments of $F_i$ which are completely off the true path. If $(i-1)$ is an integer multiple of $N_0$ then all block segments of $F_i$ will satisfy this condition. Otherwise, we will start our analysis at the next block and make the worse case assumption that all path segments up till this next block will be searched. Since the distance to the next block is at most $N_0 - 1$, this gives a maximum number of $Q^{N_0-1}$ starting blocks for any branch of $F_i$. Each $F_i$ also has $Q - 1$ branches and so this gives a generous upper bound of $(Q - 1)Q^{N_0-1}$ starting blocks for each $F_i$.

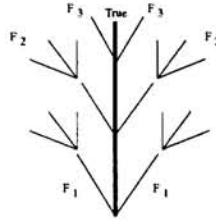

Figure 2: The target path is shown as the heavy line. The false path sets are labelled as $F_1, F_2$, etc. with the numbering depending on how soon they leave the target path. The branching factor $Q = 3$.

For each starting block, we wish to compute (or bound) the expected number of blocks that are explored thereafter. This requires computing the *fertility* of a block, the average number of paths in the block that survive pruning. Provided the fertility is smaller than one, we can then apply results from the theory of branching processes to determine the expected number of blocks searched in $F_i$.

The fertility $q$ is the number of paths that survive the geometric pruning times the probability that each survives the intensity pruning. This can be bounded (using Theorem 1) by $q \leq \hat{q}$ where:

$$\hat{q} = Q^{N_0}(N_0+1)^Q 2^{-N_0\{D(P_{\Delta g}\|U)-\hat{\epsilon}_2(\hat{T})\}}(N_0+1)^M 2^{-N_0\{D(P_{on}\|P_{off})-\epsilon_2(T)\}}$$

$$= (N_0+1)^{Q+M} 2^{-N_0\{D(P_{on}\|P_{off})-H(P_{\Delta g})-\epsilon_2(T)-\hat{\epsilon}_2(\hat{T})\}}, \quad (6)$$

where we used the fact that $D(P_{\Delta g}\|U) = \log Q - H(P_{\Delta g})$.

Observe that the condition $\hat{q} < 1$ can be satisfied provided $D(P_{on}\|P_{off})-H(P_{\Delta g}) > 0$. This condition is intuitive, it requires that the edge detector information, quantified by $D(P_{on}\|P_{off})$, must be greater than the uncertainty in the geometry measured by $H(P_{\Delta g})$. In other words, the better the edge detector and the more predictable the path geometry then the smaller $\hat{q}$ will be.

We now apply the theory of branching processes to determine the expected number of blocks explored from a starting block in $F_i$, $\sum_{z=0}^{\infty} \hat{q}^z = 1/(1-\hat{q})$. The number of branches of $F_i$ is $(Q-1)$, the total number of segments explored per block is at most $Q^{N_0}$, and we explore at most $Q^{N_0-1}$ segments before reaching the first block. The total number of $F_i$ is $N$. Therefore the total number of segments wastefully explored is at most $N(Q-1)\frac{1}{1-\hat{q}}Q^{2N_0-1}$. We summarize this result in a theorem:

**Theorem 2.** *Provided* $\hat{q} = (N_0+1)^{Q+M}2^{-N_0 K} < 1$, *where the order parameter* $K = D(P_{on}\|P_{off}) - H(P_{\Delta g}) - \epsilon_2(T) - \hat{\epsilon}_2(\hat{T})$, *then the expected number of false segments explored is at most* $N(Q-1)\frac{1}{1-\hat{q}}Q^{2N_0-1}$.

*Comment* The requirement that $\hat{q} < 1$ is chiefly determined by the *order parameter* $K = D(P_{on}\|P_{off}) - H(P_{\Delta g}) - \epsilon_2(T) - \hat{\epsilon}_2(\hat{T})$. Our convergence proof requires that $K > 0$ and will break down if $K < 0$. Is this a limitation of our proof? Or does it correspond to a fundamental difficulty in solving this tracking problem?

In more recent work (Yuille and Coughlan 1998) we extend the concept of order parameters and show that they characterize the difficulty of visual search problem *independently of the algorithm*. In other words, as $K \mapsto 0$ the problem becomes impossible to solve by any algorithm. There will be too many false paths which have better rewards than the target path. As $K \mapsto 0$ there is a phase transition in the ease of solving the problem.

## 4 Conclusion

Our analysis shows it is possible to detect certain types of image contours in linear expected time (with given starting points). We have shown how the convergence rates depend on order parameters which characterize the problem domain. In particular, the entropy of the geometric prior and the Kullback-Leibler distance between $P_{on}$ and $P_{off}$ allow us to quantify intuitions about the power of geometrical assumptions and edge detectors to solve these tasks.

Our more recent work (Yuille and Coughlan 1998) has extended this work by showing that the order parameters can be used to specify the intrinsic (algorithm independent) difficulty of the search problem and that phase transitions occur when these order parameters take critical values. In addition, we have proved convergence rates for A* algorithms which use inadmissible heuristics or combinations of heuristics and pruning (Coughlan and Yuille 1998).

As shown in (Yuille and Coughlan 1997) many of the search algorithms proposed to solve vision search problems, such as (Geman and Jedynak 1996), are special cases of A* (or close approximations). We therefore hope that the results of this paper will throw light on the success of the algorithms and may suggest practical improvements and speed ups.

### Acknowledgements

We want to acknowledge funding from NSF with award number IRI-9700446, from the Center for Imaging Sciences funded by ARO DAAH049510494, and from an ASOSRF contract 49620-98-1-0197 to ALY. We would like to thank L. Xu, D. Snow, S. Konishi, D. Geiger, J. Malik, and D. Forsyth for helpful discussions.

## References

[1] J.M. Coughlan and A.L. Yuille. "Bayesian A* Tree Search with Expected O(N) Convergence Rates for Road Tracking." Submitted to *Artificial Intelligence.* 1998.

[2] T.M. Cover and J.A. Thomas. **Elements of Information Theory**. Wiley Interscience Press. New York. 1991.

[3] D. Geman. and B. Jedynak. "An active testing model for tracking roads in satellite images". *IEEE Trans. Patt. Anal. and Machine Intel.* Vol. 18. No. 1, pp 1-14. January. 1996.

[4] J. Pearl. **Heuristics**. Addison-Wesley. 1984.

[5] A.L. Yuille and J. Coughlan. "Twenty Questions, Focus of Attention, and A*". In **Energy Minimization Methods in Computer Vision and Pattern Recognition**. Ed. M. Pellilo and E. Hancock. Springer-Verlag. (Lecture Notes in Computer Science 1223). 1997.

[6] A.L. Yuille and J.M. Coughlan. "Visual Search: Fundamental Bounds, Order Parameters, Phase Transitions, and Convergence Rates." Submitted to *Pattern Analysis and Machine Intelligence.* 1998.

[7] S.C. Zhu. "Embedding Gestalt Laws in Markov Random Fields". Submitted to *IEEE Computer Society Workshop on Perceptual Organization in Computer Vision.*
